# Odor Processing in the Bee: a Preliminary Study of the Role of Central Input to the Antennal Lobe.

**Christiane Linster**
**David Marsan**
ESPCI, Laboratoire d'Electronique
10, Rue Vauquelin, 75005 Paris
linster@neurones.espci.fr

**Claudine Masson**
Laboratoire de Neurobiologie Comparée
des Invertébrées
INRA/CNRS (URA 1190)
91140 Bures sur Yvette, France
masson@jouy.inra.fr

**Michel Kerszberg**
Institut Pasteur
CNRS (URA 1284)
Neurobiologie Moléculaire
25, Rue du Dr. Roux
75015 Paris, France

## Abstract

Based on precise anatomical data of the bee's olfactory system, we propose an investigation of the possible mechanisms of modulation and control between the two levels of olfactory information processing: the antennal lobe glomeruli and the mushroom bodies. We use simplified neurons, but realistic architecture. As a first conclusion, we postulate that the feature extraction performed by the antennal lobe (glomeruli and interneurons) necessitates central input from the mushroom bodies for fine tuning. The central input thus facilitates the evolution from *fuzzy* olfactory images in the glomerular layer towards more *focussed* images upon odor presentation.

## 1. Introduction

Honeybee foraging behavior is based on discrimination among complex odors which is the result of a memory process involving extraction and recall of "key-features" representative of the plant aroma (for a review see Masson et al. 1993). The study of the neural correlates of such mechanisms requires a determination of how the olfactory system successively analyses odors at each stage (namely: receptor cells, antennal lobe interneurons and glomeruli, mushroom bodies). Thus far, all experimental studies suggest the implication of both antennal lobe and mushroom bodies in these processes. The signal transmitted by the receptor cells is essentially unstable and fluctuating. The antennal lobe appears as the location of noise reduction and feature extraction. The specific associative components operating on the olfactory memory trace would be essentially located in the mushroom bodies. The results of neuroethological experiments indicate furthermore that both the

feed-forward connections from the antennal lobe projection neurons to the mushroom bodies and the feedback connections from the mushroom bodies to the antennal lobe neurons are crucial for the storage and the recall of odor signals (Masson 1977; Erber et al. 1980; Erber 1981).

Interestingly, the antennal lobe compares to the mammalian olfactory bulb. Computational models of the insect antennal lobe (Kerszberg and Masson 1993; Linster et al. 1993) and the mammalian olfactory bulb (Anton et al. 1991; Li and Hopfield 1989; Schild 1988) have demonstrated that feature extraction can be performed in the glomerular layer, but the possible role of central input to the glomerular layer has not been investigated (although it has been included, as a uniform signal, in the Li and Hopfield model). On the other hand, several models of the mammalian olfactory cortex (Hasselmo 1993; Wilson and Bower 1989; Liljenström 1991) have investigated its associative memory function, but have ignored the nature of the input from the olfactory bulb to this system.

Based on anatomical and electrophysiological data obtained for the bee's olfactory system (Fonta et al. 1993; Sun et al. 1993), we propose in this paper to investigate of the possible mechanisms of modulation and control between the two levels of olfactory information processing in a formal neural model. In the model, the presentation of an "odor" (a mixture of several molecules) differentially activates several populations of glomeruli. Due to coupling by local interneurons, competition is triggered between the activated glomeruli, in agreement with a recent proposal (Kerszberg and Masson 1993). We investigate the role of the different types of neurons implicated in the circuitry, and study the modulation of the glomerular states by reentrant input from the upper centers in the brain (i.e. mushroom bodies).

## 2. Olfactory circuitry in the bee's antennal lobe and mushroom bodies

95% of sensory cells located on the bee's antenna are olfactory (Esslen and Kaissling 1976), and convey signals to the antennal lobes. In the honeybee, due to some overlap of receptor cell responses, the peripheral representation of an odor stimulus is represented in an across fiber code (Fonta et al. 1993). Sensory axons project on two categories of antennal lobe neurons, namely local interneurons (LIN) and output neurons (ON). The synaptic contacts between sensory neurons and antennal lobe neurons, as well as the synaptic contacts between antennal lobe neurons are localized in areas of high synaptic density, the antennal lobe glomeruli; each glomerulus represents an *identifiable morphological neuropilar sub-unit* (of which there are 165 for the worker honeybee) (Arnold et al. 1985).

Local interneurons constitute the majority of antennal lobe neurons, and there is evidence that a majority of the LINs are inhibitory. As receptor cells are supposed to synapse mainly with LINs, the high level of excitation observed in the responses of ONs suggests that local excitation also exists (Malun 1991), in the form of spiking or non-spiking LINs, or as a modulation of local excitatbility.

All LINs are pluriglomerular, but the majority of them, heterogeneous local interneurons (or HeteroLINs), have a high density of dendrite branches in one particular glomerulus, and sparser branches distributed across other glomeruli. A second category, homogeneous local interneurons (or Homo LINs), distribute their branches more homogeneously over the whole antennal lobe. Similarly, some of the ONs have dendrites invading only one glomerulus (Uniglomerular, or Uni ON), whereas the others (Pluri ON) are pluriglomerular. The axons of both types of ON project to different areas of the protocerebrum, including the mushroom bodies (Fonta et al. 1993).

## 3. Olfactory processing in the bee's antennal lobe glomeruli

Responses of antennal lobe neurons to various odor stimuli are characterized by complex temporal patterns of activation and inactivation (Sun et al. 1993). Intracellularly recorded responses to odor mixtures are in general very complex and difficult to interpret from the responses to single odor components. A tendency to select particular odor related information is expressed by the category of "localized" antennal lobe neurons, both Hetero LINs and Uni ONs. In contrast, "global" neurons, both Homo LINs and Pluri ONs are often more responsive to mixtures than to single components. This might indicate that the related localized glomeruli represent functional sub units which are particularly involved in the discrimination of some key features.

An adaptation of the 2DG method to the honeybee antennal lobe has permitted to study the spatial distribution of odor related activity in the antennal lobe glomeruli (Nicolas et al. 1993; Masson et al. 1993). Results obtained with several individuals indicate that a correspondence can be established between two different odors and the activity maps they induce. This suggests that in the antennal lobe, different odor qualities with different biological meaning might be decoded according to separate spatial maps sharing a number of common processing areas.

## 4. Model of olfactory circuitry

In the model, we introduce the different categories of neurons described above (Figure 1). Glomeruli are grouped into several regions and each receptor cell projects onto all local interneurons with arborizations in one region. Interneurons corresponding to heterogeneous LINs can be (i) excitatory, these have a dendritic arborization (input and output synapses) restricted to one glomerulus; they provide "local" excitation, or, (ii) inhibitory, these have a dense arborization (mainly input synapses) in one glomerulus and sparse arborizations (mainly output synapses) in all others; they provide "local inhibition" and "lateral inhibition" between glomeruli. Interneurons corresponding to homogeneous LINs are inhibitory and have sparse arborizations (input and output synapses) in all glomeruli; they provide "uniform inhibition" over the glomerular layer.

Output neurons are postsynaptic only to interneurons, they do not receive direct input from receptor cells. Each output neuron collects information from all interneurons in one glomerulus: thus modeling uniglomerular ONs.

Implementation: The different neuron populations associated with one glomerulus are represented in the program as one unit (each unit is governed by one differential equation); the output of one unit represents the average firing probability of all neurons in this population (assuming that on the average, all neurons in one population receive the same input and have the same intrinsic properties). All units have membrane constants and a non-linear output function. Connection delays and connection strengths between units are chosen randomly around an average value: this assures a "realistic spatial averaging" over populations. The differential equations associated with the units are translated into difference equations and simulated by synchronous updating (sampling step 5ms).

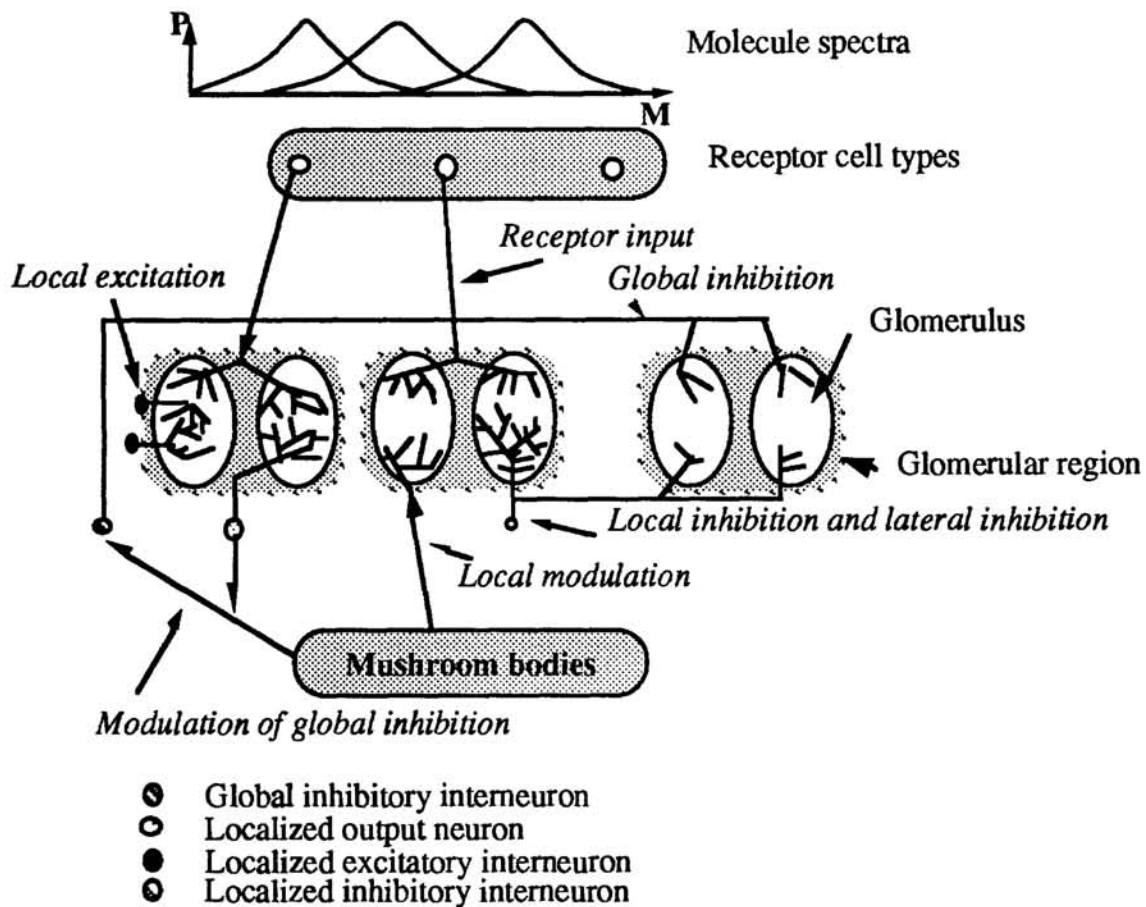

**Global inhibitory interneuron**
**Localized output neuron**
**Localized excitatory interneuron**
**Localized inhibitory interneuron**

Figure 1: Organization of the model olfactory circuitry.

In the model, we introduce receptor cells with overlapping molecule spectra; each receptor cell has its maximal spiking probability P for the presence of a particular molecule i. The axons of the receptor cells project into distinct regions of the glomerular layer. All allowed connections exist with the same probability, but with different connection strengths. The activity of each glomerulus is represented by its associated output neurons. Central input projects onto the global inhibitory interneurons (modulation of global inhibition) or on all interneurons in one glomerulus (local modulation).

## 5. Olfactory processing by the model circuitry

In the model, odors are represented as one-dimensional arrays of molecules; each molecule can be present in varying amounts. Due to the gaussian distributions of receptor cell sensitivities, an active molecule activates more than one receptor cell (with varying degrees of activation). As each receptor cell projects into all glomeruli belonging to its target region, thus, a molecular bouquet differentially activates a number of glomeruli in different glomerular regions. This triggers several phenomena: (i) due to the excitatory elements local to each glomerulus, and activated glomerulus tends to enhance the activation it receives from the receptor cells, (ii) the local inhibitory elements are activated (with a certain delay) by the receptor cell activity and by the self-activation of the local excitatory elements, and, (iii) trend to inhibit neighboring glomeruli. These phenomena result in a competition between active glomeruli: during a number of sampling steps, the output activity of each glomerulus (represented by the firing probability of the associated output neuron), oscillates from high activity to low activity. Due to the competition provided by

the lateral inhibition, the spatial oscillatory activity pattern changes over time, and a stable activity map is reached eventually. A number of glomeruli "win" and stay active, whereas others "loose" and are inhibited (Figure 2).

The activities of individual output neurons follow the general pattern described above: oscillation of the activity during a number of sampling steps until the activity "settles" down to a stable value. A stable activity can either be a constant firing probability, or a "stable" oscillation of the firing probability. An output neuron associated to a particular glomerulus may be active for a particular odor input, and silent for others. Complex temporal patterns of excitation and inhibition may occur after stimulus presentation.

Thus, the model predicts that odor representation is performed through spatial maps of activity spanning the whole glomerular layer. Individual output neurons, representing the activity of their associated glomeruli may be either excited or inhibited by a particular odor pattern.

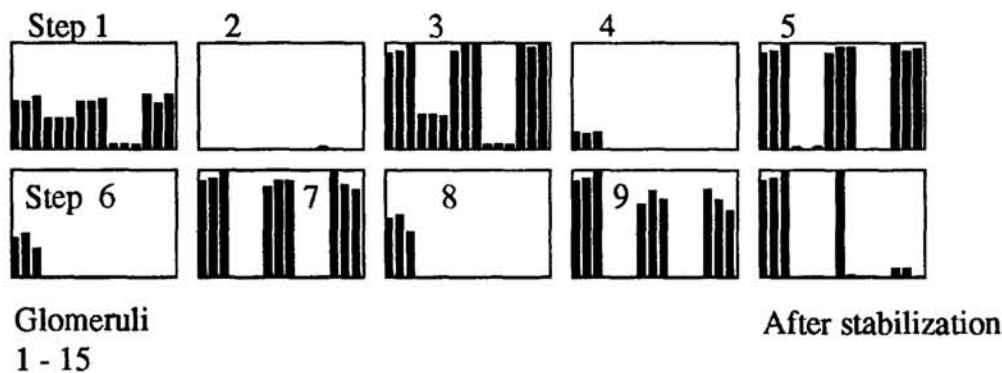

Figure 2: Behavior of the model after stimulation of the receptor cells with the molecule array indicated in the figure. For several sampling steps (of 5 ms), the activity (firing probability) of the ON associated to each glomerulus is shown. At step 1, all glomeruli are differentially activated by the receptor cell input. Lateral inhibition silences all glomeruli during the next sampling step. At step 3, some glomeruli are highly activated (due to their local excitation), whereas others are almost silenced. Then, t spatial activation pattern oscillates for a number of sampling steps (which depends on the strength of the lateral inhibitory connections and on the number of active molecules in the odor array), and finally stabilizes in a spatial activity map.

## 6. Comparison of odor processing in the Bee's antennal lobe and in the model

Antennal lobe neurons in the bee show various response patterns to stimulation with pure components and mixtures. Most LINs and ONs respond with simple excitation or inhibition to stimulation, often followed by a hyperpolarized (resp. depolarized) phase. Interestingly, most LINs respond with various degrees of excitation to stimulation with binary odors and mixtures, whereas ONs respond equally often by excitation than by inhibition (Sun et al. 1993). In the model, LINs receive direct afferent input from receptor cells, and are therefore differentially activated by odor stimulation; they respond with varying degrees of excitation to stimulation with pure components and their mixtures. Output neurons in the model receive indirect input from receptor cells *via* local interneurons. Output neurons in the model are either activated (if their associated

glomerulus *wins* the competition) or inhibited (if their associated glomerulus *looses* the competition) by odor stimulation.

In the simulations, output neurons which are excited for a particular odor stimulation belong to an active glomerulus in the spatial activity map associated to that odor. For each odor, a particular activity map is established. An output neuron is either excited or inhibited by a particular odor stimulation, indicating that it takes part in the representation of an activity map across glomeruli, which might be compared to the antennal lobe 2DG maps.

## 7. Modulation of the model dynamics

### Odor detection by modulation of spontaneous activity

At high spontaneous activity, all glomeruli in the model oscillate spontaneously (Figure 3). Odor stimulation tends to synchronize these oscillations, but no feature detection is performed. In the model, the underlying activity map which corresponds to the odor signal can only emerge if the spontaneous activity is decreased (Figure 3). Decreasing of the spontaneous activity can be achieved by 5i) activation of the global inhibitory interneurons by central input, or, (ii) decreasing of the spiking threshold of all antennal lobe neurons. These data fit well with experimental data (see Sun et al. 1993, Figures 7 and 8).

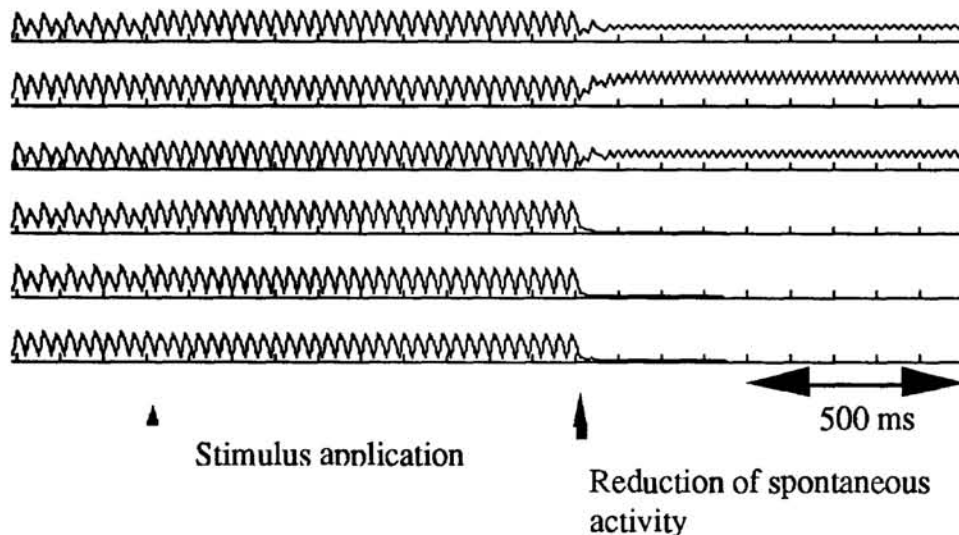

Figure 3

Figure 3: Modulation of the spontaneous activity. We show the spiking probabilities of output neurons associated to different glomeruli. Arrows indicate stimulus onset. Stimulus presentation synchronizes the oscillations. A decreasing of the spontaneous activity results in the emergence of the underlying activity map: several output neurons exhibit high activities, whereas the others are silent.

### Contrast enhancement by modulation of lateral inhibition

Presentation of an odor in the model differentially activates many or all glomeruli, which, due to the local excitation, try to enhance the activation due to the odor stimulus. Due to the competition between glomeruli, feature detection is performed in the glomerular layer, which enhances some elements of the stimulus and suppresses others.

In the model, for a given odor stimulation, the number of winning glomeruli depends on the strength of the lateral inhibition between glomeruli (Figure 5). At low lateral inhibition, most glomeruli stay active for any odor; no feature extraction is performed.

Increasing of the lateral inhibition focuses the odor maps, which can now differentiate different odor inputs.

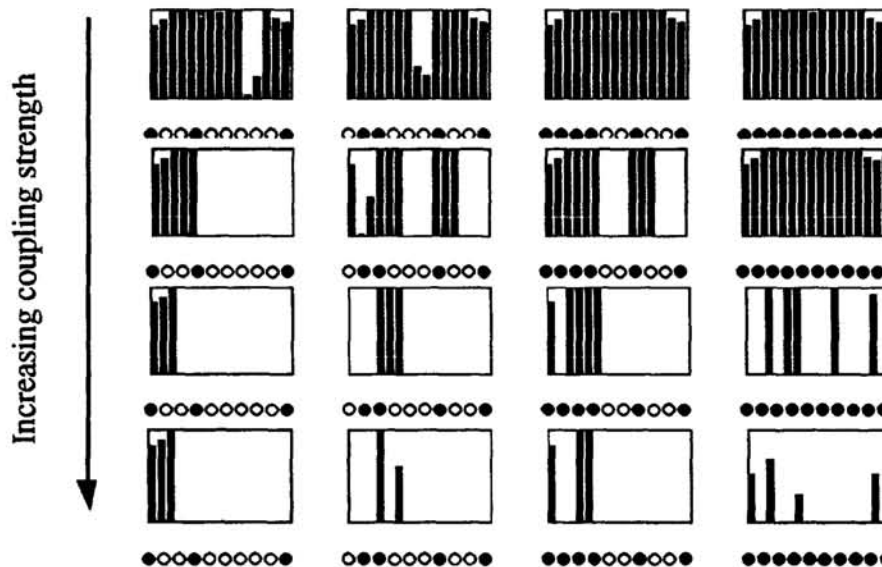

Figure 5: Stabilized activity maps for different odor stimuli with increasing lateral inhibition strength. At low competition, all glomeruli tend to be active due to their local excitation. Increasing of lateral inhibition permits to enhance the important features of each odor, and leads to uncorrelated activity maps for the different stimulations.

Increasing of the lateral inhibition permits to focus a fuzzy olfactory image in the glomerular layer, or to "smell closer". A fuzzy sampling of an odor may be useful at first approach, whereas a more precise analysis of its important components is facilitated by increasing the competition between glomeruli increases contrast enhancement.

## 8.  Discussion

We have presented the computational abilities of the neural circuitry in the antennal lobe model, based on what is known of the bee's circuitry. Single cell responses and global activity patterns are comparable to the odor processing mechanisms proposed in the insect (Linster et al. 1993; Masson et al. 1993; Kerszberg and Masson 1993) and in the vertebrate (Kauer et al. 1991; Li and Hopfield 1989; Freeman 1991) literature. As suggested by Kerszberg and Masson (1993), we show that odor preprocessing is based on spontaneous dynamics of the antennal lobe glomeruli, and that, in addition, feature detection needs competition between activated glomeruli due to global and lateral inhibition. The model is able to predict the role of the four types of neurons morphologically identified in the bee antennal lobe. It also predicts how intracellular recordings and 2DG data can be explained by the odor processing mechanism. Furthermore, modulation of the models dynamics opens up a number of new ideas about the respective role of the two main categories ("localized" and "global") of antennal lobe neurons, and the possible role of central input to these neurons.

**Acknowledgements**
The authors are grateful to G. Dreyfus and L. Personnaz for fruitful discussions.

## References

Arnold, G., Masson, C., Budhargusa, S. 1985. Comparative study of the antennal pathway of the workerbee and the drone (*Apis mellifera*). *Cell Tissue Res*. 242: 593-605.

Erber, J. 1981 Neural correlates of learning in the honeybee. *TINS* 4:270-273.

Erber, J., Masuhr, T., Menzel, R. 1980. Localisation of short-term memory in the brain of the bee, *Apis mellifera*. *Physiolo. Entomol.* 5: 343-358.

Esslen, J., Kaissling, K.E. 1976. Zahl und Verteilung antennaler Sensillen bei der Honigbiene. *Zoomorphologie* 83: 227-251.

Fonta, C., Sun, X., Masson, C. 1193. Morphology and spatial distribution of bee antennal lobe interneurons responsive to odours. *Chemical Senses* , 18 (2): pp. 101-119.

Hasselmo, M.E. 1993. Acetycholine and Learning in a Cortical Associative Memory. *Neural Computation*, 5: 32-44.

Kauer, J.S., Neff, S.R., Hamilton, K.A., Cinelli, A.R. 1991. The Salamander Olfactory Pathway: Visualizing and Modeling Circuit Activity. in *Olfaction: A Model System for Computational Neuroscience*. Davis, J. and Eichenbaum, H. (eds): 44-68. MIT Press.

Kerszberg, M., Masson, C., 1993. Signal Induced Selection among Spontaneous Activity Patterns of Bee's Olfactory Glomeruli, *submitted*.

Li, Z., Hopfield, J.J., 1989. Modeling the Olfactory Bulb and its Neural Oscillatory Processings. *Biological Cybernetics* 61:379-392.

Liljenström, H. 1991. Modeling the dynamics of olfactory cortex using simplified network units and realistic architecture. *International Journal of Neural Systems*, (1&2) : 1-15.

Linster, C., Masson, C., Kerszberg, M., Personnaz, L., Dreyfus, G. 1993 Computational Diversity in a Formal Model of the Insect Macroglomerulus., *Neural Computation*, 5:239-252.

Malun, D. 1991. Inventory and distribution of synapses of identified uniglomerular projection neurons in the antennal lobe of *periplaneta americana*. *J.Comp. Neurol.* 305: 348-360.

Masson, C. 1977. Central olfactory pathways and plasticity of responses to odor stimuli in insects. in *Olfaction and Taste* VI. Le Magnen, J., Mac Leod, P. (eds) IRL, London: 305-314.

Masson, C., Mustaparta, H. 1990. Chemical Information Processing in the Olfactory System of Insects. *Physiol. Reviews* 70(1):199-245.

Masson, C., Pham-Delègue, MH., Fonta, C., Gascuel, J., Arnold, G., Nicolas, G., Kerszberg, M. 1993. Recent advances in the concept of adaptation to natural odour signals in the honeybee *Apis mellifera* L. *Apidologie* 24: 169-194.

Menzel, R. 1983. Neurobiology of learning and memory: the honeybee as a model system. *Naturwissenschaften* 70: 504-511.

Nicolas, G., Arnold, G., Patte, F., Masson, C. 1993. Distribution régionale de l'incorporation du 3H2-Desoxyglucose dans le lobe antennaire de l'ouvrière d'abeille. *C.R. Acad. Sc. Paris (Sciences de la Vie)*, 316: 1245-1249.

Schild , D. 1988 Principles of odor coding and a neural network for odor discrimination, *Biophys. J.* 54:1001-1011.

Sun, X., Fonta, C., Masson, C. 1993. Odour quality processing by bee antennal lobe neurons. *Chemical Senses* 18 (4): 355-377.